# Prior Knowledge in Support Vector Kernels

**Bernhard Schölkopf**[*†], **Patrice Simard**[‡], **Alex Smola**[†], **& Vladimir Vapnik**[‡]
[*]Max-Planck-Institut für biologische Kybernetik, Tübingen, Germany
[†]GMD FIRST, Rudower Chaussee 5, 12489 Berlin, Germany
[‡]AT&T Research, 100 Schulz Drive, Red Bank, NJ, USA
bs@first.gmd.de

## Abstract

We explore methods for incorporating prior knowledge about a problem at hand in Support Vector learning machines. We show that both invariances under group transformations and prior knowledge about locality in images can be incorporated by constructing appropriate kernel functions.

## 1   INTRODUCTION

When we are trying to extract regularities from data, we often have additional knowledge about functions that we estimate. For instance, in image classification tasks, there exist transformations which leave class membership *invariant* (e.g. local translations); moreover, it is usually the case that images have a *local* structure in that not all correlations between image regions carry equal amounts of information.

The present study investigates the question how to make use of these two sources of knowledge by designing appropriate Support Vector (SV) kernel functions. We start by giving a brief introduction to SV machines (Vapnik & Chervonenkis, 1979; Vapnik, 1995) (Sec. 2). Regarding prior knowledge about invariances, we present a method to design kernel functions for invariant classification hyperplanes (Sec. 3). The method is applicable to invariances under the action of differentiable local 1-parameter groups of local transformations, e.g. translational invariance in pattern recognition. In Sec. 4, we describe kernels which take into account image locality by using localized receptive fields. Sec. 5 presents experimental results on both types of kernels, followed by a discussion (Sec. 6).

## 2   OPTIMAL MARGIN HYPERPLANES

For linear hyperplane decision functions $f(\mathbf{x}) = \text{sgn}\left((\mathbf{w} \cdot \mathbf{x}) + b\right)$, the VC-dimension can be controlled by controlling the norm of the weight vector $\mathbf{w}$. Given training data $(\mathbf{x}_1, y_1), \ldots, (\mathbf{x}_\ell, y_\ell)$, $\mathbf{x}_i \in \mathbf{R}^N, y_i \in \{\pm 1\}$, a separating hyperplane which generalizes

well can be found by minimizing

$$\frac{1}{2}\|\mathbf{w}\|^2 \quad \text{subject to} \quad y_i \cdot ((\mathbf{x}_i \cdot \mathbf{w}) + b) \geq 1 \quad \text{for } i = 1, \ldots, \ell, \tag{1}$$

the latter being the conditions for separating the training data with a margin. Nonseparable cases are dealt with by introducing slack variables (Cortes & Vapnik 1995), but we shall omit this modification to simplify the exposition. All of the following also applies for the nonseparable case.

To solve the above convex optimization problem, one introduces a Lagrangian with multipliers $\alpha_i$ and derives the dual form of the optimization problem: maximize

$$\sum_{i=1}^{\ell} \alpha_i - \frac{1}{2} \sum_{i,k=1}^{\ell} \alpha_i y_i \alpha_k y_k (\mathbf{x}_i \cdot \mathbf{x}_k) \quad \text{subject to} \quad \alpha_i \geq 0, \quad \sum_{i=1}^{\ell} \alpha_i y_i = 0. \tag{2}$$

It turns out that the solution vector has an expansion in terms of training examples, $\mathbf{w} = \sum_{i=1}^{\ell} \alpha_i y_i \mathbf{x}_i$, where only those $\alpha_i$ corresponding to constraints (1) which are met can become nonzero; the respective examples $\mathbf{x}_i$ are called *Support Vectors*. Substituting this expansion for $\mathbf{w}$ yields the decision function

$$f(\mathbf{x}) = \text{sgn}\left(\sum_{i=1}^{\ell} \alpha_i y_i (\mathbf{x} \cdot \mathbf{x}_i) + b\right). \tag{3}$$

It can be shown that minimizing (2) corresponds to minimizing an upper bound on the VC dimension of separating hyperplanes, or, equivalently, to maximizing the separation margin between the two classes. In the next section, we shall depart from this and modify the dot product used such that the minimization of (2) corresponds to enforcing transformation invariance, while at the same time the constraints (1) still hold.

## 3 INVARIANT HYPERPLANES

**Invariance by a self-consistency argument.** We face the following problem: to express the condition of invariance of the decision function, we already need to know its coefficients which are found only during the optimization, which in turn should already take into account the desired invariances. As a way out of this circle, we use the following ansatz: consider decision functions $f = (\text{sgn} \circ g)$, where $g$ is defined as

$$g(\mathbf{x}_j) := \sum_{i=1}^{\ell} \alpha_i y_i (B\mathbf{x}_j \cdot B\mathbf{x}_i) + b, \tag{4}$$

with a matrix $B$ to be determined below. This follows Vapnik (1995), who suggested to incorporate invariances by modifying the dot product used. Any nonsingular $B$ defines a dot product, which can equivalently be written as $(\mathbf{x}_j \cdot A\mathbf{x}_i)$, with a positive definite matrix $A = B^\top B$.

Clearly, invariance of $g$ under local transformations of all $\mathbf{x}_j$ is a sufficient condition for the local invariance of $f$, which is what we are aiming for. Strictly speaking, however, invariance of $g$ is not necessary at points which are not Support Vectors, since these lie in a region where $(\text{sgn} \circ g)$ is constant — however, before training, it is hard to predict which examples will turn out to become SVs. In the Virtual SV method (Schölkopf, Burges, & Vapnik, 1996), a first run of the standard SV algorithm is carried out to obtain an initial SV set; similar heuristics could be applied in the present case.

Local invariance of $g$ for each pattern $\mathbf{x}_j$ under transformations of a differentiable local 1-parameter group of local transformations $\mathcal{L}_t$,

$$\left.\frac{\partial}{\partial t}\right|_{t=0} g(\mathcal{L}_t \mathbf{x}_j) = 0, \tag{5}$$

can be approximately enforced by minimizing the regularizer

$$\frac{1}{\ell} \sum_{j=1}^{\ell} \left( \left. \frac{\partial}{\partial t} \right|_{t=0} g(\mathcal{L}_t \mathbf{x}_j) \right)^2. \tag{6}$$

Note that the sum may run over labelled as well as unlabelled data, so in principle one could also require the decision function to be invariant with respect to transformations of elements of a *test* set. Moreover, we could use different transformations for different patterns.

For (4), the local invariance term (5) becomes

$$\left. \cdot \frac{\partial}{\partial t} \right|_{t=0} \left( \sum_{i=1}^{\ell} \alpha_i y_i (B \mathcal{L}_t \mathbf{x}_j \cdot B \mathbf{x}_i) + b \right) = \sum_{i=1}^{\ell} \alpha_i y_i \partial_1 (B \mathcal{L}_0 \mathbf{x}_j \cdot B \mathbf{x}_i) \cdot B \left. \frac{\partial}{\partial t} \right|_{t=0} \mathcal{L}_t \mathbf{x}_j, \tag{7}$$

using the chain rule. Here, $\partial_1 (B\mathcal{L}_0\mathbf{x}_j \cdot B\mathbf{x}_i)$ denotes the gradient of $(\mathbf{x} \cdot \mathbf{y})$ with respect to $\mathbf{x}$, evaluated at the point $(\mathbf{x} \cdot \mathbf{y}) = (B\mathcal{L}_0\mathbf{x}_j \cdot B\mathbf{x}_i)$. Substituting (7) into (6), using the facts that $\mathcal{L}_0 = I$ and $\partial_1(\mathbf{x}, \mathbf{y}) = \mathbf{y}^\top$, yields the regularizer

$$\frac{1}{\ell} \sum_{j=1}^{\ell} \left( \sum_{i=1}^{\ell} \alpha_i y_i (B\mathbf{x}_i)^\top B \left. \frac{\partial}{\partial t} \right|_{t=0} \mathcal{L}_t \mathbf{x}_j \right)^2 = \sum_{i,k=1}^{\ell} \alpha_i y_i \alpha_k y_k (B\mathbf{x}_i \cdot BCB^\top B\mathbf{x}_k) \tag{8}$$

where

$$C := \frac{1}{\ell} \sum_{j=1}^{\ell} \left( \left. \frac{\partial}{\partial t} \right|_{t=0} \mathcal{L}_t \mathbf{x}_j \right) \left( \left. \frac{\partial}{\partial t} \right|_{t=0} \mathcal{L}_t \mathbf{x}_j \right)^\top. \tag{9}$$

We now choose $B$ such that (8) reduces to the standard SV target function $\|\mathbf{w}\|^2$ in the form obtained by substituting the expansion $\mathbf{w} = \sum_{i=1}^{\ell} \alpha_i y_i \mathbf{x}_i$ into it (cf. the quadratic term of (2)), utilizing the dot product chosen in (4), i.e. such that $(B\mathbf{x}_i \cdot BCB^\top B\mathbf{x}_k) = (B\mathbf{x}_i \cdot B\mathbf{x}_k)$. Assuming that the $\mathbf{x}_i$ span the whole space, this condition becomes $B^\top BCB^\top B = B^\top B$, or, by requiring $B$ to be nonsingular, i.e. that no information get lost during the preprocessing, $BCB^\top = I$. This can be satisfied by a preprocessing (whitening) matrix

$$B = C^{-\frac{1}{2}} \tag{10}$$

(modulo a unitary matrix, which we disregard), the nonnegative square root of the inverse of the nonnegative matrix $C$ defined in (9). In practice, we use a matrix

$$C_\lambda := (1 - \lambda)C + \lambda I, \tag{11}$$

$0 < \lambda \leq 1$, instead of $C$. As $C$ is nonnegative, $C_\lambda$ is invertible. For $\lambda = 1$, we recover the standard SV optimal hyperplane algorithm, other values of $\lambda$ determine the trade-off between invariance and model complexity control. It can be shown that using $C_\lambda$ corresponds to using an objective function $\Phi(\mathbf{w}) = (1 - \lambda) \sum_i (\mathbf{w} \cdot \left. \frac{\partial}{\partial t} \right|_{t=0} \mathcal{L}_t \mathbf{x}_i)^2 + \lambda \|\mathbf{w}\|^2$.

By choosing the preprocessing matrix $B$ according to (10), we have obtained a formulation of the problem where the standard SV quadratic optimization technique does in effect minimize the tangent regularizer (6): the maximum of (2), using the modified dot product as in (4), coincides with the minimum of (6) subject to the separation conditions $y_i \cdot g(\mathbf{x}_i) \geq 1$, where $g$ is defined as in (4).

Note that preprocessing with $B$ does not affect classification speed: since $(B\mathbf{x}_j \cdot B\mathbf{x}_i) = (\mathbf{x}_j \cdot B^\top B\mathbf{x}_i)$, we can precompute $B^\top B\mathbf{x}_i$ for all SVs $\mathbf{x}_i$ and thus obtain a machine (with modified SVs) which is as fast as a standard SV machine (cf. (4)).

**Relationship to Principal Component Analysis (PCA).** Let us now provide some interpretation of (10) and (9). The tangent vectors $\pm \left. \frac{\partial}{\partial t} \right|_{t=0} \mathcal{L}_t \mathbf{x}_j$ have zero mean, thus $C$ is a

sample estimate of the covariance matrix of the random vector $s \cdot \frac{\partial}{\partial t}|_{t=0} \mathcal{L}_t \mathbf{x}$, $s \in \{\pm 1\}$ being a random sign. Based on this observation, we call $C$ (9) the *Tangent Covariance Matrix* of the data set $\{\mathbf{x}_i : i = 1, \dots, \ell\}$ with respect to the transformations $\mathcal{L}_t$.

Being positive definite,[1] $C$ can be diagonalized, $C = SDS^\top$, with an orthogonal matrix $S$ consisting of $C$'s Eigenvectors and a diagonal matrix $D$ containing the corresponding positive Eigenvalues. Then we can compute $B = C^{-\frac{1}{2}} = SD^{-\frac{1}{2}}S^\top$, where $D^{-\frac{1}{2}}$ is the diagonal matrix obtained from $D$ by taking the inverse square roots of the diagonal elements. Since the dot product is invariant under orthogonal transformations, we may drop the leading $S$ and (4) becomes

$$g(\mathbf{x}_j) = \sum_{i=1}^{\ell} \alpha_i y_i (D^{-\frac{1}{2}}S^\top \mathbf{x}_j \cdot D^{-\frac{1}{2}}S^\top \mathbf{x}_i) + b. \tag{12}$$

A given pattern $\mathbf{x}$ is thus first transformed by projecting it onto the Eigenvectors of the tangent covariance matrix $C$, which are the rows of $S^\top$. The resulting feature vector is then rescaled by dividing by the square roots of $C$'s Eigenvalues.[2] In other words, the directions of main variance of the random vector $\frac{\partial}{\partial t}|_{t=0}\mathcal{L}_t\mathbf{x}$ are scaled back, thus more emphasis is put on features which are less variant under $\mathcal{L}_t$. For example, in image analysis, if the $\mathcal{L}_t$ represent translations, more emphasis is put on the relative proportions of ink in the image rather than the positions of lines. The PCA interpretation of our preprocessing matrix suggests the possibility to regularize and reduce dimensionality by discarding part of the features, as it is common usage when doing PCA.

In the present work, the ideas described in this section have only been tested in the linear case. More generally, SV machines use a nonlinear *kernel function* which can be shown to compute a dot product in a high-dimensional space $F$ nonlinearly related to input space via some map $\Phi$, i.e. $k(\mathbf{x}, \mathbf{y}) = (\Phi(\mathbf{x}) \cdot \Phi(\mathbf{y}))$. In that case, the above analysis leads to a tangent covariance matrix $C$ in $F$, and it can be shown that (12) can be evaluated in terms of the kernel function (Schölkopf, 1997). To this end, one diagonalizes $C$ using techniques of kernel PCA (Schölkopf, Smola, & Müller, 1996).

## 4   KERNELS USING LOCAL CORRELATIONS

By using a kernel $k(\mathbf{x}, \mathbf{y}) = (\mathbf{x} \cdot \mathbf{y})^d$, one implicitly constructs a decision boundary in the space of all possible products of $d$ pixels. This may not be desirable, since in natural images, correlations over short distances are much more reliable as features than long-range correlations are. To take this into account, we define a kernel $k_p^{d_1, d_2}$ as follows (cf. Fig. 1):

1. compute a third image $\mathbf{z}$, defined as the pixel-wise product of $\mathbf{x}$ and $\mathbf{y}$

2. sample $\mathbf{z}$ with pyramidal receptive fields of diameter $p$, centered at all locations $(i, j)$, to obtain the values $\mathbf{z}_{ij}$

3. raise each $\mathbf{z}_{ij}$ to the power $d_1$, to take into account local correlations within the range of the pyramid

4. sum $\mathbf{z}_{ij}^{d_1}$ over the whole image, and raise the result to the power $d_2$ to allow for longe-range correlations of order $d_2$

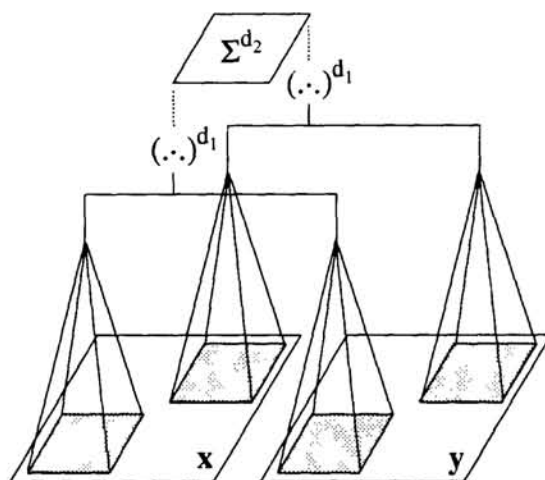

Figure 1: Kernel utilizing local correlations in images, corresponding to a dot product in a polynomial space which is spanned mainly by local correlations between pixels (see text).

The resulting kernel will be of order $d_1 \cdot d_2$, however, it will *not* contain *all* possible correlations of $d_1 \cdot d_2$ pixels.

## 5   EXPERIMENTAL RESULTS

In the experiments, we used a subset of the MNIST data base of handwritten characters (Bottou et al., 1994), consisting of 5000 training examples and 10000 test examples at a resolution of 20x20 pixels, with entries in $[-1, 1]$. Using a linear SV machine (i.e. a separating hyperplane), we obtain a test error rate of 9.8% (training 10 binary classifiers, and using the maximum value of $g$ (cf. (4)) for 10-class classification); by using a polynomial kernel of degree 4, this drops to 4.0%. In all of the following experiments, we used degree 4 kernels of various types. The number 4 was chosen as it can be written as a product of two integers, thus we could compare results to a kernel $k_p^{d_1, d_2}$ with $d_1 = d_2 = 2$. For the considered classification task, results for higher polynomial degrees are very similar.

In a series of experiments with a homogeneous polynomial kernel $k(\mathbf{x}, \mathbf{y}) = (\mathbf{x} \cdot \mathbf{y})^4$, using preprocessing with Gaussian smoothing kernels of standard deviation $0.1, 0.2, \ldots, 1.0$, we obtained error rates which gradually increased from 4.0% to 4.3%; thus no improvement of this performance was possible by a simple smoothing operation. Applying the Virtual SV method (retraining the SV machine on translated SVs; Schölkopf, Burges, & Vapnik, 1996) to this problem results in an improved error rate of 2.8%. For training on the full 60000 pattern set, the Virtual SV performance is 0.8% (Schölkopf, 1997).

**Invariant hyperplanes.** Table 1 reports results obtained by preprocessing all patterns with $B$ (cf. (10)), choosing different values of $\lambda$ (cf. (11)). In the experiments, the patterns were first rescaled to have entries in $[0, 1]$, then $B$ was computed, using horizontal and vertical translations, and preprocessing was carried out; finally, the resulting patterns were scaled back again. This was done to ensure that patterns and derivatives lie in comparable regions of $\mathbf{R}^N$ (note that if the pattern background level is a constant $-1$, then its derivative is 0). The results show that even though (9) was derived for the linear case, it can lead to improvements in the nonlinear case (here, for a degree 4 polynomial), too.

**Dimensionality reduction.** The above $[0, 1]$ scaling operation is affine rather than linear, hence the argument leading to (12) does not hold for this case. We thus only report results on dimensionality reduction for the case where the data is kept in $[0, 1]$ scaling from the very

Table 1: Classification error rates for modifying the kernel $k(\mathbf{x}, \mathbf{y}) = (\mathbf{x}\cdot\mathbf{y})^4$ with the invariant hyperplane preprocessing matrix $B_\lambda = C_\lambda^{-\frac{1}{2}}$; cf. (10) and (11). Enforcing invariance with $0.1 < \lambda < 1$ leads to improvements over the original performance ($\lambda = 1$).

| $\lambda$ | 0.1 | 0.2 | 0.3 | 0.4 | 0.5 | 0.6 | 0.7 | 0.8 | 0.9 | 1.0 |
|---|---|---|---|---|---|---|---|---|---|---|
| error rate in % | 4.2 | 3.8 | 3.6 | 3.6 | 3.7 | 3.8 | 3.8 | 3.9 | 3.9 | 4.0 |

Table 2: Dropping directions corresponding to small Eigenvalues of $C$ (cf. (12)) leads to substantial improvements. All results given are for the case $\lambda = 0.4$ (cf. Table 1); degree 4 homogeneous polynomial kernel.

| principal components discarded | 0 | 50 | 100 | 150 | 200 | 250 | 300 | 350 |
|---|---|---|---|---|---|---|---|---|
| error rate in % | 8.7 | 5.4 | 4.9 | 4.4 | 4.2 | 3.9 | 3.7 | 3.9 |

beginning on. Dropping principal components which are less important leads to substantial improvements (Table 2); cf. the explanation following (12). The results in Table 2 are somewhat distorted by the fact that the polynomial kernel is not translation invariant, and performs poorly on the $[0, 1]$ data, which becomes evident in the case where none of the principal components are discarded. Better results have been obtained using translation invariant kernels, e.g. Gaussian RBFs (Schölkopf, 1997).

**Kernels using local correlations.** To exploit locality in images, we used a pyramidal receptive field kernel $k_p^{d_1,d_2}$ with diameter $p = 9$ (cf. Sec. 4). For $d_1 = d_2 = 2$, we obtained an improved error rate of 3.1%, another degree 4 kernel with *only* local correlations ($d_1 = 4, d_2 = 1$) led to 3.4%. Albeit significantly better than the 4.0% for the degree 4 homogeneous polynomial (the error rates on the 10000 element test set have an accuracy of about 0.1%, cf. Bottou et al., 1994), this is still worse than the Virtual SV result of 2.8%. As the two methods, however, exploit different types of prior knowledge, it could be expected that combining them leads to still better performance; and indeed, this yielded the best performance of all (2.0%).

For the purpose of benchmarking, we also ran our system on the US postal service database of 7291+2007 handwritten digits at a resolution of $16 \times 16$. In that case, we obtained the following test error rates: SV with degree 4 polynomial kernel 4.2%, Virtual SV (same kernel) 3.5%, SV with $k_7^{2,2}$ 3.6%, Virtual SV with $k_7^{2,2}$ 3.0%. The latter compares favourably to almost all known results on that data base, and is second only to a memory-based tangent-distance nearest neighbour classifier at 2.6% (Simard, LeCun, & Denker, 1993).

## 6   DISCUSSION

With its rather general class of admissible kernel functions, the SV algorithm provides ample possibilities for constructing task-specific kernels. We have considered an image classification task and used two forms of domain knowledge: first, pattern classes were required to be locally translationally invariant, and second, local correlations in the images were assumed to be more reliable than long-range correlations. The second requirement can be seen as a more general form of prior knowledge — it can be thought of as arising partially from the fact that patterns possess a whole variety of transformations; in object recognition, for instance, we have object rotations and deformations. Typically, these transformations are continuous, which implies that local relationships in an image are fairly stable, whereas global relationships are less reliable.

We have incorporated both types of domain knowledge into the SV algorithm by constructing appropriate kernel functions, leading to substantial improvements on the considered pattern recognition tasks. Our method for constructing kernels for *transformation invariant* SV machines, put forward to deal with the first type of domain knowledge, so far has

only been applied in the linear case, which partially explains why it only led to moderate improvements (also, we so far only used translational invariance). It is applicable for differentiable transformations — other types, e.g. for mirror symmetry, have to be dealt with using other techniques, e.g. Virtual SVs (Schölkopf, Burges, & Vapnik, 1996). Its main advantages compared to the latter technique is that it does not slow down testing speed, and that using more invariances leaves training time almost unchanged. The proposed kernels respecting *locality* in images led to large improvements; they are applicable not only in image classification but in all cases where the relative importance of subsets of products features can be specified appropriately. They do, however, slow down both training and testing by a constant factor which depends on the specific kernel used.

Both described techniques should be directly applicable to other kernel-based methods as SV regression (Vapnik, 1995) and kernel PCA (Schölkopf, Smola, & Müller, 1996). Future work will include the nonlinear case (cf. our remarks in Sec. 3), the incorporation of invariances other than translation, and the construction of kernels incorporating local feature extractors (e.g. edge detectors) different from the pyramids described in Sec. 4.

*Acknowledgements.* We thank Chris Burges and Léon Bottou for parts of the code and for helpful discussions, and Tony Bell for his remarks.

## Footnotes

[1] It is understood that we use $C_\lambda$ if $C$ is not definite (cf. (11)). Alternatively, we can below use the pseudoinverse.

[2] As an aside, note that our goal to build invariant SV machines has thus serendipitously provided us with an approach for an open problem in SV learning, namely the one of scaling: in SV machines, there has so far been no way of automatically assigning different weight to different directions in input space — in a trained SV machine, the weights of the first layer (the SVs) form a subset of the training set. Choosing these Support Vectors from the training set only gives rather limited possibilities for appropriately dealing with different scales in different directions of input space.

# References

B. E. Boser, I .M. Guyon, and V. N. Vapnik. A training algorithm for optimal margin classifiers. In D. Haussler, editor, *Proceedings of the 5th Annual ACM Workshop on Computational Learning Theory*, pages 144–152, Pittsburgh, PA, 1992. ACM Press.

L. Bottou, C. Cortes, J. S. Denker, H. Drucker, I. Guyon, L. D. Jackel, Y. LeCun, U. A. Müller, E. Säckinger, P. Simard, and V. Vapnik. Comparison of classifier methods: a case study in handwritten digit recognition. In *Proceedings of the 12th International Conference on Pattern Recognition and Neural Networks, Jerusalem*, pages 77 – 87. IEEE Computer Society Press, 1994.

C. Cortes and V. Vapnik. Support vector networks. *Machine Learning*, 20:273 – 297, 1995.

B. Schölkopf. *Support Vector Learning*. R. Oldenbourg Verlag, Munich, 1997. ISBN 3-486-24632-1.

B. Schölkopf, C. Burges, and V. Vapnik. Incorporating invariances in support vector learning machines. In C. von der Malsburg, W. von Seelen, J. C. Vorbrüggen, and B. Sendhoff, editors, *Artificial Neural Networks — ICANN'96*, pages 47 – 52, Berlin, 1996a. Springer Lecture Notes in Computer Science, Vol. 1112.

B. Schölkopf, A. Smola, and K.-R. Müller. Nonlinear component analysis as a kernel eigenvalue problem. Technical Report 44, Max-Planck-Institut für biologische Kybernetik, 1996b. in press *(Neural Computation)*.

P. Simard, Y. LeCun, and J. Denker. Efficient pattern recognition using a new transformation distance. In S. J. Hanson, J. D. Cowan, and C. L. Giles, editors, *Advances in Neural Information Processing Systems 5*, pages 50–58, San Mateo, CA, 1993. Morgan Kaufmann.

P. Simard, B. Victorri, Y. LeCun, and J. Denker. Tangent prop — a formalism for specifying selected invariances in an adaptive network. In J. E. Moody, S. J. Hanson, and R. P. Lippmann, editors, *Advances in Neural Information Processing Systems 4*, pages 895–903, San Mateo, CA, 1992. Morgan Kaufmann.

V. Vapnik. *The Nature of Statistical Learning Theory*. Springer Verlag, New York, 1995.

V. Vapnik and A. Chervonenkis. *Theory of Pattern Recognition [in Russian]*. Nauka, Moscow, 1974. (German Translation: W. Wapnik & A. Tscherwonenkis, *Theorie der Zeichenerkennung*, Akademie-Verlag, Berlin, 1979).